# Nonlinear Discriminant Analysis using Kernel Functions

**Volker Roth & Volker Steinhage**
University of Bonn, Institut of Computer Science III
Römerstrasse 164, D-53117 Bonn, Germany
*{roth, steinhag}@cs.uni-bonn.de*

## Abstract

Fishers linear discriminant analysis (LDA) is a classical multivariate technique both for dimension reduction and classification. The data vectors are transformed into a low dimensional subspace such that the class centroids are spread out as much as possible. In this subspace LDA works as a simple prototype classifier with linear decision boundaries. However, in many applications the linear boundaries do not adequately separate the classes. We present a nonlinear generalization of discriminant analysis that uses the *kernel trick* of representing dot products by kernel functions. The presented algorithm allows a simple formulation of the EM-algorithm in terms of kernel functions which leads to a unique concept for unsupervised mixture analysis, supervised discriminant analysis and semi-supervised discriminant analysis with partially unlabelled observations in feature spaces.

## 1   Introduction

Classical linear discriminant analysis (LDA) projects $N$ data vectors that belong to $c$ different classes into a $(c-1)$–dimensional space in such way that the ratio of *between group scatter* $S_B$ and *within group scatter* $S_W$ is maximized [1]. LDA formally consists of an eigenvalue decomposition of $S_W^{-1} S_B$ leading to the so called *canonical variates* which contain the whole class specific information in a $(c-1)$-dimensional subspace. The canonical variates can be ordered by decreasing eigenvalue size indicating that the first variates contain the major part of the information. As a consequence, this procedure allows low dimensional representations and therefore a *visualization* of the data. Besides from interpreting LDA only as a technique for dimensionality reduction, it can also be seen as a multi-class classification method: the set of linear discriminant functions define a partition of the projected space into regions that are identified with class membership. A new observation $x$ is assigned to the class with centroid closest to $x$ in the projected space.

To overcome the limitation of only linear decision functions some attempts have been made to incorporate nonlinearity into the classical algorithm. HASTIE *et al.* [2] introduced the so called model of *Flexible Discriminant Analysis*: LDA is reformulated in the framework of linear regression estimation and a generalization of this method is given by using nonlinear regression techniques. The proposed regression techniques implement the idea of using nonlinear mappings to transform the input data into a new space in which again a linear regression is performed. In real world

applications this approach has to deal with numerical problems due to the dimensional explosion resulting from nonlinear mappings. In the recent years approaches that avoid such *explicit* mappings by using *kernel functions* have become popular. The main idea is to construct algorithms that only afford dot products of pattern vectors which can be computed efficiently in high-dimensional spaces. Examples of this type of algorithms are the *Support Vector Machine* [3] and *Kernel Principal Component Analysis* [4].

In this paper we show that it is possible to formulate classical linear regression and therefore also linear discriminant analysis exclusively in terms of dot products. Therefore, kernel methods can be used to construct a nonlinear variant of discriminant analysis. We call this technique *Kernel Discriminant Analysis* (KDA). Contrary to a similar approach that has been published recently [5], our algorithm is a real multi-class classifier and inherits from classical LDA the convenient property of data *visualization*.

## 2    Review of Linear Discriminant Analysis

Under the assumption of the data being centered (i.e. $\sum_i x_i = 0$) the *scatter matrices* $S_B$ and $S_W$ are defined by

$$S_B = \sum_{j=1}^{c} \frac{1}{n_j} \sum_{l,m=1}^{n_j} \left( x_l^{(j)} \right) \left( x_m^{(j)} \right)^T \tag{1}$$

$$S_W = \sum_{j=1}^{c} \sum_{l=1}^{n_j} \left( x_l^{(j)} - \frac{1}{n_j} \sum_{l=1}^{n_j} x_l^{(j)} \right) \left( x_l^{(j)} - \frac{1}{n_j} \sum_{m=1}^{n_j} x_m^{(j)} \right)^T, \tag{2}$$

where $n_j$ is the number of patterns $x_l^{(j)}$ that belong to class $j$.

LDA chooses a transformation matrix $V$ that maximizes the objective function

$$J(V) = \frac{|V^T S_B V|}{|V^T S_W V|}. \tag{3}$$

The columns of an optimal $V$ are the generalized eigenvectors that correspond to the nonzero eigenvalues in $S_B v_i = \lambda_i S_W v_i$.

In [6] and [7] we have shown, that the standard LDA algorithm can be restated exclusively in terms of dot products of input vectors. The final equation is an eigenvalue equation in terms of dot product matrices which are of size $N \times N$. Since the solution of high-dimensional generalized eigenvalue equations may cause numerical problems ($N$ may be large in real world applications), we present an improved algorithm that reformulates discriminant analysis as a regression problem. Moreover, this version allows a simple implementation of the EM-algorithm in feature spaces.

## 3    Linear regression analysis

In this section we give a brief review of linear regression analysis which we use as "building block" for LDA. The task of linear regression analysis is to approximate the regression function by a linear function

$$r(x) = E(\mathcal{Y}|\mathcal{X} = x) \approx c + x^T \beta. \tag{4}$$

on the basis of a sample $(y_1, x_1), \cdots, (y_N, x_N)$. Let now $y$ denote the vector $(y_1, \ldots, y_N)^T$ and $X$ denote the data matrix which rows are the input vectors. Using a quadratic loss function, the optimal parameters $c$ and $\beta$ are chosen to minimize the average squared residual

$$ASR = N^{-1} \left\| y - c \, 1_N + X\beta \right\|^2 + \beta^T \Omega \beta. \tag{5}$$

$1_N$ denotes a $N$-vector of ones, $\Omega$ denotes a ridge-type penalty matrix $\Omega = \epsilon I$ which penalizes the coefficients of $\beta$. Assuming the data being centered, i.e $\sum_{i=1}^{N} x_i = 0$, the parameters of the regression function are given by:

$$c = N^{-1} \sum_{i=1}^{N} y_i =: \mu_y, \qquad \beta = (X^T X + \epsilon I)^{-1} X^T y. \tag{6}$$

## 4  LDA by optimal scoring

In this section the LDA problem is linked to linear regression using the framework of *penalized optimal scoring*. We give an overview over the detailed derivation in [2] and [8]. Considering again the problem with $c$ classes and $N$ data vectors, the class-memberships are represented by a categorical response variable $\mathcal{G}$ with $c$ levels. It is useful to code the $n$ responses in terms of the indicator matrix $Z$: $Z_{i,j} = 1$, if the $i$-th data vector belongs to class j, and 0 otherwise. The point of optimal scoring is to turn categorical variables into quantitative ones by assigning scores to classes: the score vector $\boldsymbol{\theta}$ assigns the real number $\boldsymbol{\theta}_j$ to the $j$-th level of $\mathcal{G}$. The vector $Z\boldsymbol{\theta}$ then represents a vector of scored training data and is regressed onto the data matrix $X$. The simultaneous estimation of scores and regression coefficients constitutes the optimal scoring problem: minimize the criterion

$$ASR(\boldsymbol{\theta},\boldsymbol{\beta}) = N^{-1}\big[\|Z\boldsymbol{\theta} - X\boldsymbol{\beta}\|^2 + \boldsymbol{\beta}^T\Omega\boldsymbol{\beta}\big] \tag{7}$$

under the constraint $\frac{1}{N}\|Z\boldsymbol{\theta}\|^2 = 1$. According to (6), for a given score $\boldsymbol{\theta}$ the minimizing $\boldsymbol{\beta}$ is given by

$$\boldsymbol{\beta}_{OS} = (X^TX + \Omega)^{-1}X^TZ\boldsymbol{\theta}, \tag{8}$$

and the partially minimized criterion becomes:

$$\min_{\boldsymbol{\beta}} ASR(\boldsymbol{\theta},\boldsymbol{\beta}) = 1 - N^{-1}\boldsymbol{\theta}^T Z^T M(\Omega) Z\boldsymbol{\theta}, \tag{9}$$

where $M(\Omega) = X(X^TX + \Omega)^{-1}X^T$ denotes the regularized *hat* or *smoother* matrix. Minimizing of (9) under the constraint $\frac{1}{N}\|Z\boldsymbol{\theta}\|^2 = 1$ can be performed by the following procedure:

1. Choose an initial matrix $\Theta_0$ satisfying the constraint $N^{-1}\Theta_0^T Z^T Z\Theta_0 = I$ and set $\Theta_0^* = Z\Theta_0$
2. Run a multi-response regression of $\Theta_0^*$ onto $X$: $\widehat{\Theta}_0^* = M(\Omega)\Theta_0^* = XB$, where $B$ is the matrix of regression coefficients.
3. Eigenanalyze $\Theta_0^{*T}\widehat{\Theta}_0^*$ to obtain the optimal scores, and update the matrix of regression coefficients: $B^* = BW$, with $W$ being the matrix of eigenvectors.

It can be shown, that the final matrix $B^*$ is, up to a diagonal scale matrix, equivalent to the matrix of LDA-vectors, see [8].

## 5  Ridge regression using only dot products

The penalty matrix $\Omega$ in (5) assures that the penalized $d \times d$ covariance matrix $\tilde{\Sigma} = X^TX + \epsilon I$ is a symmetric nonsingular matrix. Therefore, it has $d$ eigenvectors $\boldsymbol{e}_i$ with accomplished positive eigenvalues $\gamma_i$ such that the following equations hold:

$$\tilde{\Sigma}\boldsymbol{e}_i = \sum_{j=1}^{N} \boldsymbol{x}_j\boldsymbol{x}_j^T\boldsymbol{e}_i + \epsilon\boldsymbol{e}_i = \gamma_i\boldsymbol{e}_i, \qquad \tilde{\Sigma}^{-1} = \sum_{i=1}^{d} \frac{1}{\gamma_i}\boldsymbol{e}_i\boldsymbol{e}_i^T \tag{10}$$

The first equation implies that the first $l$ leading eigenvectors $\boldsymbol{e}_i$ with eigenvalues $\gamma_i > \epsilon$ have an expansion in terms of the input vectors. Note that $l$ is the number of nonzero eigenvalues of the unpenalized covariance matrix $X^TX$. Together with (6), it follows for the general case, when the dimensionality $d$ may extend $l$, that $\boldsymbol{\beta}$ can be written as the sum of two terms: an expansion in terms of the vectors $\boldsymbol{x}_i$ with coefficients $\alpha_i$ and a similar expansion in terms of the remaining eigenvectors:

$$\boldsymbol{\beta} = \sum_{i=1}^{N} \alpha_i\boldsymbol{x}_i + \sum_{j=l+1}^{d} \xi_j\boldsymbol{e}_j = X^T\boldsymbol{\alpha} + \sum_{j=l+1}^{d} \xi_j\boldsymbol{e}_j, \tag{11}$$

with $\boldsymbol{\alpha} = (\alpha_1 \cdots \alpha_n)^T$. However, the last term can be dropped, since every eigenvector $\boldsymbol{e}_j$, $j = l+1,\ldots,d$ is orthogonal to *every* vector $\boldsymbol{x}_i$ and does not influence the value of the regression function (4).
The problem of penalized linear regression can therefore be stated as minimizing

$$ASR(\boldsymbol{\alpha}) = N^{-1}\big[\,\|\boldsymbol{y} - XX^T\boldsymbol{\alpha}\|^2 + \boldsymbol{\alpha}^T X\Omega X^T\boldsymbol{\alpha}\,\big]. \tag{12}$$

A stationary vector $\boldsymbol{\alpha}$ is determined by
$$\boldsymbol{\alpha} = (XX^T + \Omega)^{-1}\boldsymbol{y}. \tag{13}$$

Let now the *dot product matrix* $K$ be defined by $K_{ij} = \boldsymbol{x}_i^T\boldsymbol{x}_j$ and let for a given test point $(\boldsymbol{x}_l)$ the dot product vector $\boldsymbol{k}_l$ be defined by $\boldsymbol{k}_l = X\boldsymbol{x}_l$. With this notation the regression function of a test point $(\boldsymbol{x}_l)$ reads

$$r(\boldsymbol{x}_l) = \mu_y + \boldsymbol{k}_l^T\left(K + \epsilon I\right)^{-1}\boldsymbol{y}. \tag{14}$$

This equation requires only dot products and we can apply the *kernel trick*. The final equation (14), up to the constant term $\mu_y$, has also been found by SAUNDERS et al., [9]. They restated ridge regression in dual variables and optimized the resulting criterion function with a lagrange multiplier technique. Note that our derivation, which is a direct generalization of the standard linear regression formalism, leads in a natural way to a class of more general regression functions including the constant term.

## 6  LDA using only dot products

Setting $\boldsymbol{\beta} = X^T\boldsymbol{\alpha}$ as in (11) and using the notation of section 5, for a given score $\boldsymbol{\theta}$ the optimal vector $\boldsymbol{\alpha}$ is given by:
$$\boldsymbol{\alpha}_{OS} = (XX^T + \Omega)^{-1}Z\boldsymbol{\theta}. \tag{15}$$

Analogous to (9), the partially minimized criterion becomes:
$$\min_{\boldsymbol{\alpha}} ASR(\boldsymbol{\theta}, \boldsymbol{\alpha}) = 1 - N^{-1}\boldsymbol{\theta}^T Z^T \tilde{M}(\Omega)Z\boldsymbol{\theta}, \tag{16}$$

with
$$\tilde{M}(\Omega) = XX^T(XX^T + \Omega)^{-1} = K(K + \epsilon I)^{-1}.$$

To minimize (16) under the constraint $\frac{1}{N}\|Z\boldsymbol{\theta}\|^2 = 1$ the procedure described in section 4 can be used when $M(\Omega)$ is substituted by $\tilde{M}(\Omega)$. The matrix $Y$ which rows are the input vectors projected onto the column vectors of $B^*$ is given by:
$$Y = XB^* = K(K + \epsilon I)^{-1}Z\Theta_0 W. \tag{17}$$

Note that again the dot product matrix $K$ is all that is needed to calculate $Y$.

## 7  The kernel trick

The main idea of constructing nonlinear algorithms is to apply the linear methods not in the space of observations but in a *feature space* $\boldsymbol{F}$ that is related to the former by a nonlinear mapping $\phi : \boldsymbol{R}^N \to \boldsymbol{F}$, $\boldsymbol{x} \to \phi(\boldsymbol{x})$.
Assuming that the mapped data are centered in $\boldsymbol{F}$, i.e. $\sum_{i=1}^n \phi(\boldsymbol{x}_i) = \boldsymbol{0}$, the presented algorithms remain formally unchanged if the dot product matrix $K$ is computed in $\boldsymbol{F}$: $K_{ij} = (\phi(\boldsymbol{x}_i) \cdot \phi(\boldsymbol{x}_j))$. As shown in [4], this assumption can be dropped by writing $\tilde{\phi}$ instead of the mapping $\phi$: $\tilde{\phi}(\boldsymbol{x}_i) := \phi(\boldsymbol{x}_i) - \frac{1}{n}\sum_{i=1}^n \phi(\boldsymbol{x}_i)$.
Computation of dot products in feature spaces can be done efficiently by using *kernel functions* $k(\boldsymbol{x}_i, \boldsymbol{x}_j)$ [3]: For some choices of $k$ there exists a mapping $\phi$ into some feature space $\boldsymbol{F}$ such that $k$ acts as a dot product in $\boldsymbol{F}$. Among possible kernel functions there are e.g. Radial Basis Function (RBF) kernels of the form $k(\boldsymbol{x}, \boldsymbol{y}) = \exp(-\|\boldsymbol{x} - \boldsymbol{y}\|^2/c)$.

## 8  The EM-algorithm in feature spaces

LDA can be derived as the maximum likelihood method for normal populations with different means and common covariance matrix $\Sigma$ (see [11]). Coding the class membership of the observations in the matrix $Z$ as in section 4, LDA maximizes the (complete data) log-likelihood function

$$l(\mu_k, \Sigma, \pi_k) \propto - \sum_{k=1}^{c} \sum_{Z_{ik}=1} \log\left((x_i - \mu_k)^T \Sigma^{-1} (x_i - \mu_k)\right) - N \log|\Sigma|. \quad (18)$$

This concept can be generalized for the case that only the group membership of $N_C < N$ observations is known ([14], p.679): the EM-algorithm provides a convenient method for maximizing the likelihood function with missing data:

**E-step:** set $p_{ki} = \text{Prob}(x_i \in \text{class } k)$

$$p_{ki} = \begin{cases} Z_{ik}, & \text{if the class membership of } x_i \text{ has been observed} \\ \frac{\pi_k \phi_k(x_i)}{\sum_{k=1}^{c} \pi_k \phi_k(x_i)}, & \text{otherwise,} \quad \phi_k(x_i) \propto \exp[-1/2(x_i - \mu_k)^T \Sigma^{-1}(x_i - \mu_k)] \end{cases}$$

**M-step:** set

$$\pi_k = \frac{1}{N} \sum_{i=1}^{N} p_{ki}, \quad \mu_k = \frac{1}{N\pi_k} \sum_{i=1}^{N} p_{ki} x_i, \quad \Sigma = \frac{1}{N} \sum_{k=1}^{c} \sum_{i=1}^{N} p_{ki} (x_i - \mu_k)(x_i - \mu_k)^T$$

The idea behind this approach is that even an unclassified observation can be used for estimation if it is given a proper weight according to its posterior probability for class membership. The M-step can be seen as weighted mean and covariance maximum likelihood estimates in a weighted and augmented problem: we augment the data by replicating the $N$ observations $c$ times, with the $l$-th such replication having observation weights $p_{li}$. The maximization of the likelihood function can be achieved via a weighted and augmented LDA. It turns out that it is *not* necessary to explicitly replicate the observations and run a standard LDA: the optimal scoring version of LDA described in section 4 allows an implicit solution of the augmented problem that still uses only $N$ observations. Instead of using a response indicator matrix $Z$, one uses a *blurred* response Matrix $\tilde{Z}$, whose rows consist of the current class probabilities for each observation. At each M-step this $\tilde{Z}$ is used in a multiple linear regression followed by an eigen-decomposition. A detailed derivation is given in [11]. Since we have shown that the optimal scoring problem can be solved in feature spaces using kernel functions this is also the case for the whole EM-algorithm: the E-step requires only *differences* in Mahalonobis distances which are supplied by KDA.

After iterated application of the E- and M-step an observation is classified to the class $k$ with highest probability $p_k$. This leads to a unique framework for pure mixture analysis ($N_C = 0$), pure discriminant analysis ($N_C = N$) and the semi-supervised models of discriminant analysis with partially unclassified observations ($0 < N_C < N$) in feature spaces.

## 9  Experiments

**Waveform data**: We illustrate KDA on a popular simulated example, taken from [10], p.49-55 and used in [2, 11]. It is a three class problem with 21 variables. The learning set consisted of 100 observations per class. The test set was of size 1000. The results are given in table 1.

Table 1: Results for waveform data. The values are averages over 10 simulations. The 4 entries above the line are taken from [11]. QDA: quadratic discriminant analysis, FDA: flexible discriminant analysis, MDA: mixture discriminant analysis.

| Technique | Training Error [%] | Test Error [%] |
|---|---|---|
| LDA | 12.1(0.6) | 19.1(0.6) |
| QDA | 3.9(0.4) | 20.5(0.6) |
| FDA (best model parameters) | 10.0(0.6) | 19.1(0.6) |
| MDA (best model parameters) | 13.9(0.5) | 15.5(0.5) |
| KDA (RBF kernel, $\sigma = 2, \epsilon = 1.5$) | 10.7(0.6) | 14.1(0.7) |

The Bayes risk for the problem is about 14% [10]. KDA outperforms the other nonlinear versions of discriminant analysis and reaches the Bayes rate within the error bounds, indicating that one cannot expect significant further improvement using other classifiers. Figure 1 demonstrates the *data visualization* property of KDA. Since for a 3 class problem the dimensionality of the projected space equals 2, the data can be visualized without any loss of information. In the left plot one can see the projected learn data and the class centroids, the right plot shows the test data and again the class centroids of the learning set.

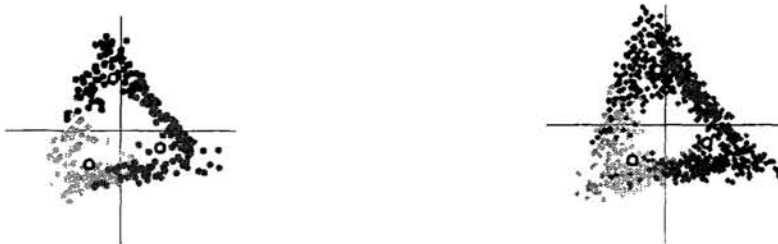

Figure 1: Data visualization with KDA. Left: learn set, right: test set

To demonstrate the effect of using unlabeled data for classification we repeated the experiment with waveform data using only 20 labeled observations per class. We compared the the classification results on a test set of size 300 using only the labeled data (error rate $E_1$) with the results of the EM-model which considers the test data as incomplete measurements during an iterative maximization of the likelihood function (error rate $E_2$). Using a RBF kernel ($\sigma = 250$), we obtained the following mean error rates over 20 simulations: $E_1 = 30.5(3.6)\%$, $E_2 = 17.1(2.7)\%$. The classification performance could be drastically improved when including the unlabelled data into the learning process.

**Object recognition:** We tested KDA on the *MPI Chair Database*[1]. It consists of 89 regular spaced views form the upper viewing hemisphere of 25 different classes of chairs as a training set and 100 random views of each class as a test set. The available images are downscaled to $16 \times 16$ pixels. We did not use the additional 4 edge detection patterns for each view. Classification results for several classifiers are given in table 2.

Table 2: Test error rates (%). Support Vector Machine , Multi Layer Perceptron, Oriented Filter, taken from [12].

| SVM | MLP | OF | KDA, RBF kernel | KDA poly. kernel |
|---|---|---|---|---|
| 2.0 | 7.2 | 21.0 | 1.9 | 2.1 |

For a comparison of the computational performance we also trained the *SVM-light* implementation (V 2.0) on the data, [13]. In this experiment with 25 classes the KDA algorithm showed to be significantly faster than the SVM: using the RBF-kernel, KDA was 3 times faster, with the polynomial kernel KDA was 20 times faster than *SVM-light*.

## 10   Discussion

In this paper we present a nonlinear version of classical linear discriminant analysis. The main idea is to map the input vectors into a high- or even infinite dimensional feature space and to apply LDA in this enlarged space. Restating LDA in a way that only dot products of input vectors are needed makes it possible to use *kernel representations* of dot products. This overcomes numerical problems in high-dimensional

feature spaces. We studied the classification performance of the KDA classifier on simulated waveform data and on the MPI chair database that has been widely used for benchmarking in the literature. For medium size problems, especially if the number of classes is high, the KDA algorithm showed to be significantly faster than a SVM while leading to the same classification performance. From classical LDA the presented algorithm inherits the convenient property of *data visualization*, since it allows low dimensional views of the data vectors. This makes an intuitive interpretation possible, which is helpful in many practical applications. The presented KDA algorithm can be used as the maximization step in an EM algorithm in feature spaces. This allows to include unlabeled observation into the learning process which can improve classification results. Studying the performance of KDA for other classification problems as well as a theoretical comparison of the optimization criteria used in the KDA- and SVM-algorithm will be subject of future work.

## Acknowledgements
This work was supported by Deutsche Forschungsgemeinschaft, DFG. We heavily profitted from discussions with Armin B. Cremers, John Held and Lothar Hermes.

## Footnotes

[1]The database is available via ftp://ftp.mpik-tueb.mpg.de/pub/chair_dataset/

## References

[1] R. Duda and P. Hart, *Pattern Classification and Scene Analysis*. Wiley & Sons, 1973.

[2] T. Hastie, R. Tibshirani, and A. Buja, "Flexible discriminant analysis by optimal scoring," *JASA*, vol. 89, pp. 1255–1270, 1994.

[3] V. N. Vapnik, *Statistical learning theory*. Wiley & Sons, 1998.

[4] B. Schölkopf, A. Smola, and K.-R. Muller, "Nonlinear component analysis as a kernel eigenvalue problem," *Neural Computation*, vol. 10, no. 5, pp. 1299–1319, 1998.

[5] S. Mika, G. Rätsch, J. Weston, B. Schölkopf, and K.-R. Müller, "Fisher discriminant analysis with kernels," in *Neural Networks for Signal Processing IX* (Y.-H. Hu, J. Larsen, E. Wilson, and S. Douglas, eds.), pp. 41–48, IEEE, 1999.

[6] V. Roth and V. Steinhage, "Nonlinear discriminant analysis using kernel functions," Tech. Rep. IAI-TR-99-7, Department of Computer Science III, Bonn University, 1999.

[7] V. Roth, A. Pogoda, V. Steinhage, and S. Schröder, "Pattern recognition combining feature- and pixel-based classification within a real world application," in *Mustererkennung 1999* (W. Förstner, J. Buhmann, A. Faber, and P. Faber, eds.), Informatik aktuell, pp. 120-129, 21. DAGM Symposium, Bonn, Springer, 1999.

[8] T. Hastie, A. Buja, and R. Tibshirani, "Penalized discriminant analysis," *AnnStat*, vol. 23, pp. 73–102, 1995.

[9] S. Saunders, A. Gammermann, and V. Vovk, "Ridge regression learning algorithm in dual variables," tech. rep., Royal Holloway, University of London, 1998.

[10] L. Breiman, J. H. Friedman, R. A. Olshen, and C. J. Stone, *Classification and Regression Trees*. Monterey, CA: Wadsworth and Brooks/Cole, 1984.

[11] T. Hastie and R. Tibshirani, "Discriminant analysis by gaussian mixtures," *JRSSB*, vol. 58, pp. 158–176, 1996.

[12] B. Schölkopf, *Support Vector Learning*. PhD thesis, 1997. R. Oldenbourg Verlag, Munich.

[13] T. Joachims, "Making large-scale svm learning practical," in *Advances in Kernel Methods – Support Vector Learning* (B. Schölkopf, C. Burges, and A. Smola, eds.), MIT Press, 1999.

[14] B. Flury, *A First Course in Multivariate Statistics*. Springer, 1997.